# Boosting with Multi-Way Branching in Decision Trees

Yishay Mansour        David McAllester

AT&T Labs-Research
180 Park Ave
Florham Park NJ 07932
{mansour, dmac}@research.att.com

## Abstract

It is known that decision tree learning can be viewed as a form of boosting. However, existing boosting theorems for decision tree learning allow only binary-branching trees and the generalization to multi-branching trees is not immediate. Practical decision tree algorithms, such as CART and C4.5, implement a trade-off between the number of branches and the improvement in tree quality as measured by an index function. Here we give a boosting justification for a particular quantitative trade-off curve. Our main theorem states, in essence, that if we require an improvement proportional to the log of the number of branches then top-down greedy construction of decision trees remains an effective boosting algorithm.

## 1 Introduction

Decision trees have been proved to be a very popular tool in experimental machine learning. Their popularity stems from two basic features — they can be constructed quickly and they seem to achieve low error rates in practice. In some cases the time required for tree growth scales linearly with the sample size. Efficient tree construction allows for very large data sets. On the other hand, although there are known theoretical handicaps of the decision tree representations, it seem that in practice they achieve accuracy which is comparable to other learning paradigms such as neural networks.

While decision tree learning algorithms are popular in practice it seems hard to quantify their success in a theoretical model. It is fairly easy to see that even if the target function can be described using a small decision tree, tree learning algorithms may fail to find a good approximation. Kearns and Mansour [6] used the weak learning hypothesis to show that standard tree learning algorithms perform boosting. This provides a theoretical justification for decision tree learning similar

to justifications that have been given for various other boosting algorithms, such as AdaBoost [4].

Most decision tree learning algorithms use a top-down growth process. Given a current tree the algorithm selects some leaf node and extends it to an internal node by assigning to it some "branching function" and adding a leaf to each possible output value of this branching function. The set of branching functions may differ from one algorithm to another, but most algorithms used in practice try to keep the set of branching functions fairly simple. For example, in C4.5 [7], each branching function depends on a single attribute. For categorical attributes, the branching is according to the attribute's value, while for continuous attributes it performs a comparison of the attribute with some constant.

Of course such top-down tree growth can over-fit the data — it is easy to construct a (large) tree whose error rate on the training data is zero. However, if the class of splitting functions has finite VC dimension then it is possible to prove that, with high confidence of the choice of the training data, for all trees $T$ the true error rate of $T$ is bounded by $\hat{\epsilon}(T) + O\left(\sqrt{|T|/m}\right)$ where $\hat{\epsilon}(T)$ is the error rate of $T$ on the training sample, $|T|$ is the number of leaves of $T$, and $m$ is the size of the training sample. Over-fitting can be avoided by requiring that top-down tree growth produce a small tree. In practice this is usually done by constructing a large tree and then pruning away some of its nodes. Here we take a slightly different approach. We assume a given target tree size $s$ and consider the problem of constructing a tree $T$ with $|T| = s$ and $\hat{\epsilon}(T)$ as small as possible. We can avoid over-fitting by selecting a small target value for the tree size.

A fundamental question in top-down tree growth is how to select the branching function when growing a given leaf. We can think of the target size as a "budget". A four-way branch spends more of the tree size budget than does a two-way branch — a four-way branch increases the tree size by roughly the same amount as two two-way branches. A sufficiently large branch would spend the entire tree size budget in a single step. Branches that spend more of the tree size budget should be required to achieve more progress than branches spending less of the budget. Naively, one would expect that the improvement should be required to be roughly linear in the number of new leaves introduced — one should get a return proportional to the expense. However, a weak learning assumption and a target tree size define a nontrivial game between the learner and an adversary. The learner makes moves by selecting branching functions and the adversary makes moves by presenting options consistent with the weak learning hypothesis. We prove here that the learner achieve a better value in this game by selecting branches that get a return considerably smaller than the naive linear return. Our main theorem states, in essence, that the return need only be proportional to the log of the number of branches.

## 2 Preliminaries

We assume a set $\mathcal{X}$ of instances and an unknown target function $f$ mapping $\mathcal{X}$ to $\{0, 1\}$. We assume a given "training set" $S$ which is a set of pairs of the form $\langle x, f(x) \rangle$. We let $\mathcal{H}$ be a set of potential branching functions where each $h \in \mathcal{H}$ is a function from $\mathcal{X}$ to a finite set $R_h$ — we allow different functions in $\mathcal{H}$ to have different ranges. We require that for any $h \in \mathcal{H}$ we have $|R_h| \geq 2$. An $\mathcal{H}$-tree is

a tree where each internal node is labeled with an branching function $h \in \mathcal{H}$ and has children corresponding to the elements of the set $R_h$. We define $|T|$ to be the number of leaf nodes of $T$. We let $L(T)$ be the set of leaf nodes of $T$. For a given tree $T$, leaf node $\ell$ of $T$ and sample $S$ we write $S_\ell$ to denote the subset of the sample $S$ reaching leaf $\ell$. For $\ell \in T$ we define $\hat{p}_\ell$ to be the fraction of the sample reaching leaf $\ell$, i.e., $|S_\ell|/|S|$. We define $\hat{q}_\ell$ to be the fraction of the pairs $\langle x, f(x) \rangle$ in $S_\ell$ for which $f(x) = 1$. The training error of $T$, denoted $\hat{\epsilon}(T)$, is $\sum_{\ell \in L(T)} \hat{p}_\ell \min(\hat{q}_\ell, 1 - \hat{q}_\ell)$.

## 3   The Weak Learning Hypothesis and Boosting

Here, as in [6], we view top-down decision tree learning as a form of Boosting [8, 3]. Boosting describes a general class of iterative algorithms based on a weak learning hypothesis. The classical weak learning hypothesis applies to classes of Boolean functions. Let $\mathcal{H}_2$ be the subset of branching functions $h \in \mathcal{H}$ with $|R_h| = 2$. For $\delta > 0$ the classical $\delta$-weak learning hypothesis for $\mathcal{H}_2$ states that for any distribution on $\mathcal{X}$ there exists an $h \in \mathcal{H}_2$ with $Pr_D(h(x) \neq f(x)) \leq 1/2 - \delta$. Algorithms designed to exploit this particular hypothesis for classes of Boolean functions have proved to be quite useful in practice [5].

Kearns and Mansour show [6] that the key to using the weak learning hypothesis for decision tree learning is the use of an index function $I : [0,1] \rightarrow [0,1]$ where $I(q) \leq 1$, $I(q) \geq \min(q, (1 - q))$ and where $I(T)$ is defined to be $\sum_{\ell \in L(T)} \hat{p}_\ell I(\hat{q}_\ell)$. Note that these conditions imply that $\hat{\epsilon}(T) \leq I(T)$. For any sample $W$ let $q_W$ be the fraction of pairs $\langle x, f(x) \rangle \in W$ such that $f(x) = 1$. For any $h \in \mathcal{H}$ let $T_h$ be the decision tree consisting of a single internal node with branching function $h$ plus a leaf for each member of $|R_h|$. Let $I_W(T_h)$ denote the value of $I(T_h)$ as measured with respect to the sample $W$. Let $\Delta(W, h)$ denote $I(q_W) - I_W(T_h)$. The quantity $\Delta(W, h)$ is the reduction in the index for sample $W$ achieved by introducing a single branch. Also note that $\hat{p}_\ell \Delta(S_\ell, h)$ is the reduction in $I(T)$ when the leaf $\ell$ is replaced by the branch $h$. Kearns and Mansour [6] prove the following lemma.

**Lemma 3.1 (Kearns & Mansour)** *Assuming the $\delta$-weak learning hypothesis for $\mathcal{H}_2$, and taking $I(q)$ to be $2\sqrt{q(1-q)}$, we have that for any sample $W$ there exists an $h \in \mathcal{H}_2$ such that $\Delta(W, h) \geq \frac{\delta^2}{16} I(q_W)$.*

This lemma motivates the following definition.

**Definition 1** *We say that $\mathcal{H}_2$ and $I$ satisfies the $\gamma$-weak tree-growth hypothesis if for any sample $W$ from $\mathcal{X}$ there exists an $h \in \mathcal{H}_2$ such that $\Delta(W, h) \geq \gamma I(q_W)$.*

Lemma 3.1 states, in essence, that the classical weak learning hypothesis implies the weak tree growth hypothesis for the index function $I(q) = 2\sqrt{q(1-q)}$. Empirically, however, the weak tree growth hypothesis seems to hold for a variety of index functions that were already used for tree growth prior to the work of Kearns and Mansour. The Ginni index $I(q) = 4q(1 - q)$ is used in CART [1] and the entropy $I(q) = -q \log q - (1 - q) \log(1 - q)$ is used in C4.5 [7]. It has long been empirically observed that it is possible to make steady progress in reducing $I(T)$ for these choices of $I$ while it is difficult to make steady progress in reducing $\hat{\epsilon}(T)$.

We now define a simple binary branching procedure. For a given training set $S$ and target tree size $s$ this algorithm grows a tree with $|T| = s$. In the algorithm

$\emptyset$ denotes the trivial tree whose root is a leaf node and $T_{\ell,h}$ denotes the result of replacing the leaf $\ell$ with the branching function $h$ and a new leaf for each element of $R_h$.

$T = \emptyset$
**WHILE** $(|T| < s)$ **DO**
$\qquad \ell \leftarrow \text{argmax}_\ell \ \hat{p}_\ell I(\hat{q}_\ell)$
$\qquad h \leftarrow \text{argmax}_{h \in \mathcal{H}_2} \Delta(S_\ell, h)$
$\qquad T \leftarrow T_{\ell,h};$
**END-WHILE**

We now define $e(n)$ to be the quantity $\prod_{i=1}^{n-1}(1 - \frac{\gamma}{n})$. Note that $e(n) \le \prod_{i=1}^{n-1} e^{-\frac{\gamma}{i}} = e^{-\gamma \sum_{i=1}^{n-1} 1/i} \le e^{-\gamma \ln n} = n^{-\gamma}$.

**Theorem 3.2 (Kearns & Mansour)** *If $\mathcal{H}_2$ and $I$ satisfy the $\gamma$-weak tree growth hypothesis then the binary branching procedure produces a tree $T$ with $\hat{e}(T) \le I(T) \le e(|T|) \le |T|^{-\gamma}$.*

**Proof:** The proof is by induction on the number of iterations of the procedure. We have that $I(\emptyset) \le 1 = e(1)$ so the initial tree immediately satisfies the condition. We now assume that the condition is satisfied by $T$ at the begining of an iteration and prove that it remains satisfied by $T_{\ell,h}$ at the end of the iteration. Since $I(T) = \sum_{\ell \in T} \hat{p}_\ell I(\hat{q}_\ell)$ we have that the leaf $\ell$ selected by the procedure is such that $\hat{p}_\ell I(\hat{q}_\ell) \ge \frac{I(T)}{|T|}$. By the $\gamma$-weak tree growth assumption the function $h$ selected by the procedure has the property that $\Delta(S_\ell, h) \ge \gamma I(\hat{q}_\ell)$. We now have that $I(T) - I(T_{\ell,h}) = \hat{p}_\ell \Delta(S_\ell, \ h) \ge \hat{p}_\ell \gamma I(\hat{q}_\ell) \ge \gamma \frac{I(T)}{|T|}$. This implies that $I(T_{\ell,h}) \le I(T) - \frac{\gamma}{|T|} I(T) = (1 - \frac{\gamma}{T}) I(T) \le (1 - \frac{\gamma}{|T|}) e(|T|) = e(|T| + 1) = e(|T_{\ell,h}|)$.
$\square$

# 4  Statement of the Main Theorem

We now construct a tree-growth algorithm that selects multi-way branching functions. As with many weak learning hypotheses, the $\gamma$-weak tree-growth hypothesis can be viewed as defining a game between the learner and an adversary. Given a tree $T$ the adversary selects a set of branching functions allowed at each leaf of the tree subject to the constraint that at each leaf $\ell$ the adversary must provide a binary branching function $h$ with $\Delta(S_\ell, h) \ge \gamma I(\hat{q}_\ell)$. The learner then selects a leaf $\ell$ and a branching function $h$ and replaces $T$ by $T_{\ell,h}$. The adversary then again selects a new set of options for each leaf subject to the $\gamma$-weak tree growth hypothesis. The proof of theorem 3.2 implies that even when the adversary can reassign all options at every move there exists a learner strategy, the binary branching procedure, guaranteed to achieves a final error rate of $|T|^{-\gamma}$.

Of course the optimal play for the adversary in this game is to only provide a single binary option at each leaf. However, in practice the "adversary" will make mistakes and provide options to the learner which can be exploited to achieve even lower error rates. Our objective now is to construct a strategy for the learner which can exploit multi-way branches provided by the adversary.

We first say that a branching function $h$ is *acceptable* for tree $T$ and target size

$s$ if either $|R_h| = 2$ or $|T| < e(|R_h|)s\gamma/(2|R_h|)$. We also define $g(k)$ to be the quantity $(1 - e(k))/\gamma$. It should be noted that $g(2) = 1$. It should also be noted that $e(k) \sim e^{-\gamma \ln k}$ and hence for $\gamma \ln k$ small we have $e(k) \sim 1 - \gamma \ln k$ and hence $g(k) \sim \ln k$. We now define the following multi-branch tree growth procedure.

$T = \emptyset$
**WHILE** $(|T| < s)$ **DO**
       $\ell \leftarrow \text{argmax}_\ell \ \hat{p}_\ell I(\hat{q}_\ell)$
       $h \leftarrow \text{argmax}_{h \in \mathcal{H}, \ h \text{ acceptable for } T \text{ and } s} \ \ \Delta(S_\ell, h)/g(|R_h|)$
       $T \leftarrow T_{\ell,h};$
**END-WHILE**

A run of the multi-branch tree growth procedure will be called $\gamma$-boosting if at each iteration the branching function $h$ selected has the property that $\Delta(S_\ell, h)/g(|R_h|) \geq \gamma I(\hat{q}_\ell)$. The $\gamma$-weak tree growth hypothesis implies that $\Delta(S_\ell, h)/g(|R_h|) \geq \gamma I(\hat{q}_\ell)/g(2) = \gamma I(\hat{q}_\ell)$. Therefore, the $\gamma$-weak tree growth hypothesis implies that every run of the multi-branch growth procedure is $\gamma$-bootsing. But a run can be $\gamma$-bootsing by exploiting mutli-way branches even when the $\gamma$-weak tree growth hypothesis fails. The following is the main theorem of this paper.

**Theorem 4.1** *If $T$ is produced by a $\gamma$-boosting run of the multi-branch tree-growth procedure then $I(T) \leq e(|T|) \leq |T|^{-\gamma}$.*

## 5  Proof of Theorem 4.1

To prove the main theorem we need the concept of a visited weighted tree, or VW-tree for short. A VW-tree is a tree in which each node $m$ is assigned both a rational weight $w_m \in [0,1]$ and an integer visitation count $v_m \geq 1$. We now define the following VW tree growth procedure. In the procedure $T_w$ is the tree consisting of a single root node with weight $w$ and visitation count 1. The tree $T_{\ell,w_1,\dots,w_k}$ is the result of inserting $k$ new leaves below the leaf $\ell$ where the $i$th new leaf has weight $w_i$ and new leaves have visitation count 1.

$w \leftarrow$ any rational number in $[0,1]$
$T \leftarrow T_w$
**FOR ANY NUMBER OF STEPS REPEAT THE FOLLOWING**
       $\ell \leftarrow \text{argmax}_\ell \ \frac{e(v_\ell)w_\ell}{v_\ell}$
       $v_\ell \leftarrow v_\ell + 1$
       **OPTIONALLY** $T \leftarrow T_{\ell,w_1,\dots,w_{v_\ell}}$ **WITH** $w_1 + \dots w_{v_\ell} \leq e(v_\ell)w_\ell$

We first prove an analog of theorem 3.2 for the above procedure. For a VW-tree $T$ we define $|T|$ to be $\sum_{\ell \in L(T)} v_\ell$ and we define $I(T)$ to be $\sum_{\ell \in L(T)} e(v_\ell)w_\ell$.

**Lemma 5.1** *The VW procedure maintains the invariant that $I(T) \leq e(|T|)$.*

**Proof:** The proof is by induction on the number of iterations of the algorithm. The result is immediate for the initial tree since $e(1) = 1$. We now assume that $I(T) \leq e(|T|)$ at the start of an iteration and show that this remains true at the end of the iteration.

We can associate each leaf $\ell$ with $v_\ell$ "subleaves" each of weight $e(v_\ell)w_\ell/v_\ell$. We have that $|T|$ is the total number of these subleaves and $I(T)$ is the total weight of these subleaves. Therefore there must exist a subleaf whose weight is at least $I(T)/|T|$. Hence there must exist a leaf $\ell$ satisfying $e(v_\ell)w_\ell/v_\ell \geq I(T)/|T|$. Therefore this relation must hold of the leaf $\ell$ selected by the procedure.

Let $T'$ be the tree resulting from incrementing $v_\ell$. We now have $I(T) - I(T') = e(v_\ell)w_\ell - e(v_\ell + 1)w_\ell = e(v_\ell)w_\ell - (1 - \frac{\gamma}{v_\ell})e(v_\ell)w_\ell = \frac{\gamma}{v_\ell}e(v_\ell)w_\ell \geq \gamma\frac{I(T)}{|T|}$. So we have $I(T') \leq (1 - \frac{\gamma}{|T|})I(T) \leq (1 - \frac{\gamma}{|T|})e(|T|) = e(|T'|)$.

Finally, if the procedure grows new leaves we have that the $I(T)$ does not increase and that $|T|$ remains the same and hence the invariant is maintained. $\square$

For any internal node $m$ in a tree $T$ let $C(m)$ denote the set of nodes which are children of $m$. A VW-tree will be called *locally-well-formed* if for every internal node $m$ we have that $v_m = |C(m)|$, that $\sum_{n \in C(m)} w_n \leq e(|C(m)|)w_m$. A VW-tree will be called *globally-safe* if $\max_{\ell \in L(T)} e(v_\ell)w_\ell/v_\ell \leq \min_{m \in N(T)} e(v_\ell - 1)w_\ell/(v_\ell - 1)$ where $N(T)$ denotes the set of internal nodes of $T$.

**Lemma 5.2** *If $T$ is a locally well-formed and globally safe VW-tree, then $T$ is a possible output of the VW growth procedure and therefore $I(T) \leq e(|T|)$.*

**Proof:** Since $T$ is locally well formed we can use $T$ as a "template" for making nondeterministic choices in the VW growth procedure. This process is guaranteed to produce $T$ provided that the growth procedure is never forced to visit a node corresponding to a leaf of $T$. But the global safety condition guarantees that any unfinished internal node of $T$ has a weight as least as large as any leaf node of $T$. $\square$

We now give a way of mapping $\mathcal{H}$-trees into VW-trees. More specifically, for any $\mathcal{H}$-tree $T$ we define $VW(T)$ to be the result of assigning each node $m$ in $T$ the weight $\hat{p}_m I(\hat{q}_m)$, each internal node a visitation count equal to its number of children, and each leaf node a visitation count equal to 1. We now have the following lemmas.

**Lemma 5.3** *If $T$ is grown by a $\gamma$-boosting run of the multi-branch procedure then $VW(T)$ is locally well-formed.*

**Proof:** Note that the children of an internal node $m$ are derived by selecting a branching function $h$ for the node $m$. Since the run is $\gamma$-boosting we have $\Delta(S_\ell, h)/g(|R_h|) \geq \gamma I(\hat{q}_\ell)$. Therefore $\Delta(S_\ell, h) = (I(\hat{q}_\ell) - I_{S_\ell}(T_h)) \geq I(\hat{q}_\ell)(1 - e(|R_h|))$. This implies that $I_{S_\ell}(T_h) \leq e(|R_h|)I(\hat{q}_\ell)$. Multiplying by $\hat{p}_\ell$ and transforming the result into weights in the tree $VW(T)$ gives the desired result. $\square$

The following lemma now suffices for theorem 4.1.

**Lemma 5.4** *If $T$ is grown by a $\gamma$-boosting run of the multi-branch procedure then $VW(T)$ is globally safe.*

**Proof:** First note that the following is an invariant of a $\gamma$-boosting run of the multi-branch procedure.

$$\max_{\ell \in L(VW(T))} w_\ell \leq \min_{m \in N(VW(T))} w_\ell$$

The proof is a simple induction on $\gamma$-boosting tree growth using the fact that the procedure always expands a leaf node of maximal weight.

We must now show that for every internal node $m$ and every leaf $\ell$ we have that $w_\ell \leq e(k-1)w_m/(k-1)$ where $k$ is the number of children of $m$. Note that if $k = 2$ then this reduces to $w_\ell \leq w_m$ which follows from the above invariant. So we can assume without loss of generality that $k > 2$. Also, since $e(k)/k < e(k-1)/(k-1)$, it suffices to show that $w_\ell \leq e(k)w_m/k$.

Let $m$ be an internal node with $k > 2$ children and let $T'$ be the tree at the time $m$ was selected for expansion. Let $w_\ell$ be the maximum weight of a leaf in the final tree $T$. By the definition of the acceptability condition, in the last $s/2$ iterations we are performing only binary branching. Each binary expansion reduces the index by at least $\gamma$ times the weight of the selected node. Since the sequence of nodes selected in the multi-branch procedure has non-increasing weights, we have that in any iteration the weight of the selected node is at least $w_\ell$. Since there are at least $s/2$ binary expansions after the expansion of $m$, each of which reduces $I$ by at least $\gamma w_\ell$, we have that $s\gamma w_\ell/2 \leq I(T')$ so $w_\ell \leq 2I(T')/(\gamma s)$. The acceptability condition can be written as $2/(\gamma s) \leq e(k)/(k|T'|)$ which now yields $w_l \leq I(T')e(k)/(k|T'|)$. But we have that $I(T')/|T'| \leq w_m$ which now yields $w_l \leq e(k)w_m/k$ as desired. $\square$

# References

[1] Leo Breiman, Jerome H. Friedman, Richard A. Olshen, and Charles J. Stone. *Classification and Regression Trees*. Wadsworth International Group, 1984.

[2] Tom Dietterich, Michael Kearns and Yishay Mansour. Applying the Weak Learning Framework to understand and improve C4.5. In Proc. of *Machine Learning*, 96-104, 1996.

[3] Yoav Freund. Boosting a weak learning algorithm by majority. *Information and Computation*, 121(2):256–285, 1995.

[4] Yoav Freund and Robert E. Schapire. A decision-theoretic generalization of on-line learning and an application to boosting. In *Computational Learning Theory: Second European Conference, EuroCOLT '95*, pages 23–37. Springer-Verlag, 1995.

[5] Yoav Freund and Robert E. Schapire. Experiments with a new boosting algorithm. In *Machine Learning: Proceedings of the Thirteenth International Conference*, pages 148–156, 1996.

[6] Michael Kearns and Yishay Mansour. On the boosting ability of top-down decision tree learning. In *Proceedings of the Twenty-Eighth ACM Symposium on the Theory of Computing*, pages 459–468, 1996.

[7] J. Ross Quinlan. *C4.5: Programs for Machine Learning*. Morgan Kaufmann, 1993.

[8] Robert E. Schapire. The strength of weak learnability. *Machine Learning*, 5(2):197–227, 1990.